# HARMONET: A Neural Net for Harmonizing Chorales in the Style of J.S.Bach

**Hermann Hild**
hhild@ira.uka.de

**Johannes Feulner**
johannes@ira.uka.de

**Wolfram Menzel**
menzel@ira.uka.de

Institut für Logik, Komplexität und Deduktionssysteme
Am Fasanengarten 5
Universität Karlsruhe
W-7500 Karlsruhe 1, Germany

## Abstract

HARMONET, a system employing connectionist networks for music processing, is presented. After being trained on some dozen Bach chorales using error backpropagation, the system is capable of producing four-part chorales in the style of J.S.Bach, given a one-part melody. Our system solves a musical real-world problem on a performance level appropriate for musical practice. HARMONET's power is based on (a) a new coding scheme capturing musically relevant information and (b) the integration of backpropagation and symbolic algorithms in a hierarchical system, combining the advantages of both.

## 1   INTRODUCTION

Neural approaches to music processing have been previously proposed (Lischka, 1989) and implemented (Mozer, 1991)(Todd, 1989). The promise neural networks offer is that they may shed some light on an aspect of human creativity that doesn't seem to be describable in terms of symbols and rules. Ultimately what music is (or isn't) lies in the eye (or ear) of the beholder. The great composers, such as Bach or Mozart, learned and obeyed quite a number of rules, e.g. the famous prohibition of parallel fifths. But these rules alone do not suffice to characterize a personal or even historic style. An easy test is to generate music at random, using only

Figure 1: The beginning of the chorale melody "Jesu, meine Zuversicht" and its harmonization by J.S.Bach

schoolbook rules as constraints. The result is "error free" but aesthetically offensive. To overcome this gap between obeying rules and producing music adhering to an accepted aesthetic standard, we propose HARMONET, which integrates symbolic algorithms and neural networks to compose four part chorales in the style of J.S. Bach (1685 - 1750), given the one part melody. The neural nets concentrate on the creative part of the task, being responsible for aesthetic conformance to the standard set by Bach in nearly 400 examples. Original Bach Chorales are used as training data. Conventional algorithms do the bookkeeping tasks like observing pitch ranges, or preventing parallel fifths. HARMONET's level of performance approaches that of improvising church organists, making it applicable to musical practice.

## 2   TASK DEFINITION

The process of composing an accompaniment for a given chorale melody is called **chorale harmonization**. Typically, a chorale melody is a plain melody often harmonized to be sung by a choir. Correspondingly, the four voices of a chorale harmonization are called soprano (the melody part), alto, tenor and bass. Figure 1 depicts an example of a chorale melody and its harmonization by J.S.Bach. For centuries, music students have been routinely taught to solve the task of chorale harmonization. Many theories and rules about "dos" and "don'ts" have been developed. However, the task of HARMONET is to learn to harmonize chorales *from example*. Neural nets are used to find stylisticly characteristic harmonic sequences and ornamentations.

# 3   SYSTEM OVERVIEW

Given a set of Bach chorales, our goal is to find an approximation $\hat{f}$ of the quite complex function[1] $f$ which maps chorale melodies into their harmonization as demonstrated by J.S.Bach on almost 400 examples. In the following sections we propose a decomposition of $f$ into manageable subfunctions.

## 3.1   TASK DECOMPOSITION

The learning task is decomposed along two dimensions:

**Different levels of abstractions.** The **chord skeleton** is obtained if eighth and sixteenth notes are viewed as omitable ornamentations. Furthermore, if the chords are conceived as harmonies with certain attributes such as "inversion" or "characteristic dissonances", the chorale is reducible to its **harmonic skeleton**, a thoroughbass-like representation (Figure 2).

**Locality in time.** The accompaniment is divided into smaller parts, each of which is learned independently by looking at some local context, a window. Treating small parts independently certainly hurts global consistency. Some of the dependencies lost can be regained if the current decision window additionally considers the outcome of its predecessors (external feedback). Figure 3 shows two consecutive windows cut out from the harmonic skeleton.

To harmonize a chorale, HARMONET starts by learning the harmonic skeleton, which then is refined to the chord skeleton and finally augmented with ornamenting quavers (Figure 4, left side).

## 3.2   THE HARMONIC SKELETON

Chorales have a rich harmonic structure, which is mainly responsible for their "musical appearance". Thus generating a good harmonic skeleton is the most important of HARMONET's subtasks. HARMONET creates a harmonic sequence by sweeping through the chorale melody and determining a harmony for each quarter note, considering its local context and the previously found harmonies as input.
At each quarterbeat position $t$, the following information is extracted to form one training example:

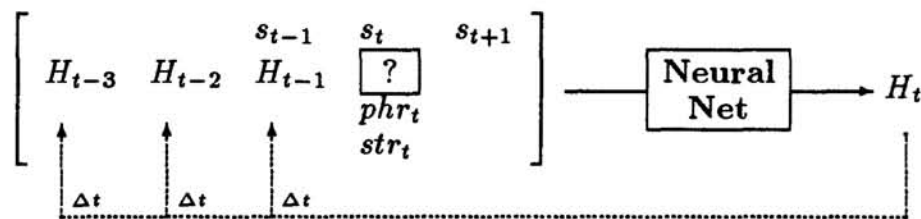

The target to be learned (the harmony $H_t$ at position $t$) is marked by the box. The input consists of the harmonic context to the left (the external feedback $H_{t-3}$, $H_{t-2}$ and $H_{t-1}$) and the melodic context (pitches $s_{t-1}$, $s_t$ and $s_{t+1}$). $phr_t$ contains

Figure 2: The chord and the harmonic skeleton of the chorale from figure 1.

information about the relative position of $t$ to the beginning or end of a musical phrase. $str_t$ is a boolean value indicating whether $s_t$ is a stressed quarter. A harmony $H_t$ has three components: Most importantly, the *harmonic function* relates the key of the harmony to the key of the piece. The *inversion* indicates the bass note of the harmony. The *characteristic dissonances* are notes which do not directly belong to the harmony, thus giving it additional tension.

The **coding of pitch** is decisive for recognizing musically relevant regularities in the training examples. This problem is discussed in many places (Shepard, 1982) (Mozer, 1991). We developed a new coding scheme guided by the harmonic necessities of homophonic music pieces: A note $s$ is represented as the set of harmonic functions that contain $s$, as shown below:

| Fct. | T | D | S | Tp | Sp | Dp | DD | DP | TP | d | Vtp | SS |
|------|---|---|---|----|----|----|----|----|----|---|-----|----|
| C    | 1 | 0 | 1 | 1  | 0  | 0  | 0  | 0  | 0  | 0 | 0   | 0  |
| D    | 0 | 1 | 0 | 0  | 1  | 0  | 1  | 0  | 0  | 1 | 1   | 1  |
| E    | .. |   |   |    |    |    |    |    |    |   |     |    |

**T, D, S, Tp** etc. are standard musical abbreviations to denote harmonic functions. The resulting representation is distributed with respect to pitch. However, it is local with respect to harmonic functions. This allows the network to anticipate future harmonic developments even though there cannot be a lookahead for harmonies yet uncomposed.

Besides the 12 input units for each of the pitches $s_{t-1}$, $s_t$, $s_{t+1}$, we need $12+5+3 =$

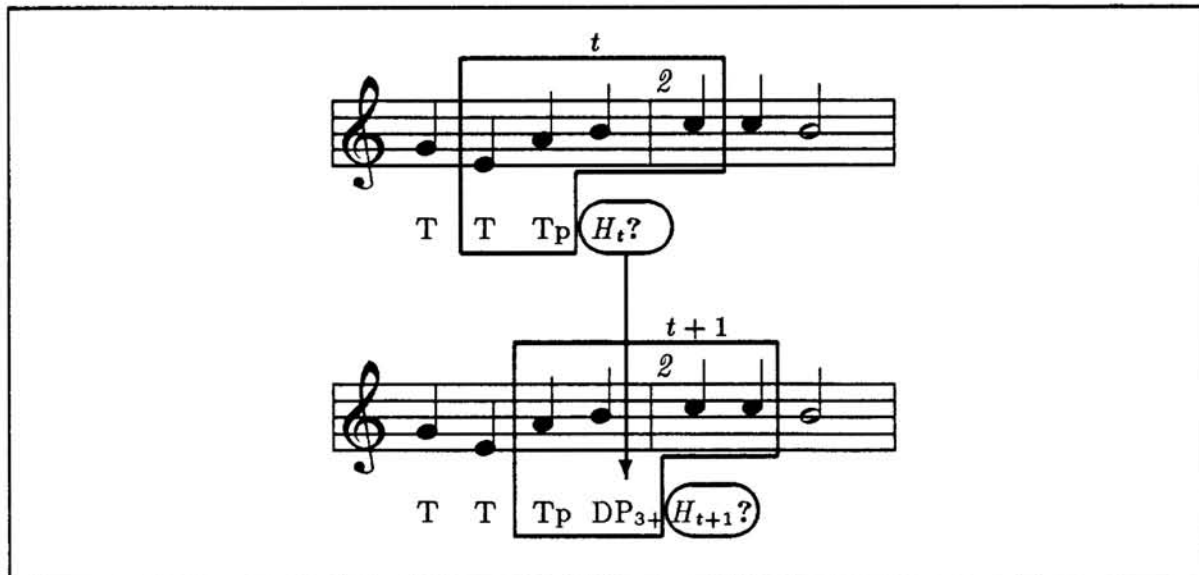

Figure 3: The harmonic skeleton broken into local windows. The harmony $H_t$, determined at quarterbeat position $t$, becomes part of the input of the window at position $t + 1$.

20 input units for each of the 3 components of the harmonies $H_{t-3}$, $H_{t-2}$ and $H_{t-1}$, 9 units to code the phrase information $phr_t$ and 1 unit for the stress $str_t$. Thus our net has a total of $3 * 12 + 3 * 20 + 9 + 1 = 106$ input units and 20 output units. We used one hidden layer with 70 units.

In a more advanced version (Figure 4, right side), we use three nets (N1, N2, N3) in parallel, each of which was trained on windows of different size. The harmonic function for which the majority of these three nets votes is passed to two subsequent nets (N4, N5) determining the chord inversion and characteristic dissonances of the harmony. Using windows of different sizes in parallel employs statistical information to solve the problem of chosing an appropriate window size.

## 3.3   THE CHORD SKELETON

The task on this level is to find the two middle parts (alto and tenor) given the soprano $S$ of the chorale melody and the harmony $H$ determined by the neural nets. Since $H$ includes information about the chord inversion, the pitch of the bass (modulo its octave) is already given. The problem is tackled with a "generate and test" approach: Symbolic algorithms select a "best" chord out of the set of all chords consistent with the given harmony $H$ and common chorale constraints.

## 3.4   QUAVER ORNAMENTATIONS

In the last subtask, another net is taught how to add ornamenting eighths to the chord skeleton. The output of this network is the set of eighth notes (if any) by which a particular chord $C_t$ can be augmented. The network's input describes the local context of $C_t$ in terms of attributes such as the intervals between $C_t$ and $C_{t+1}$, voice leading characteristics, or the presence of eighths in previous chords.

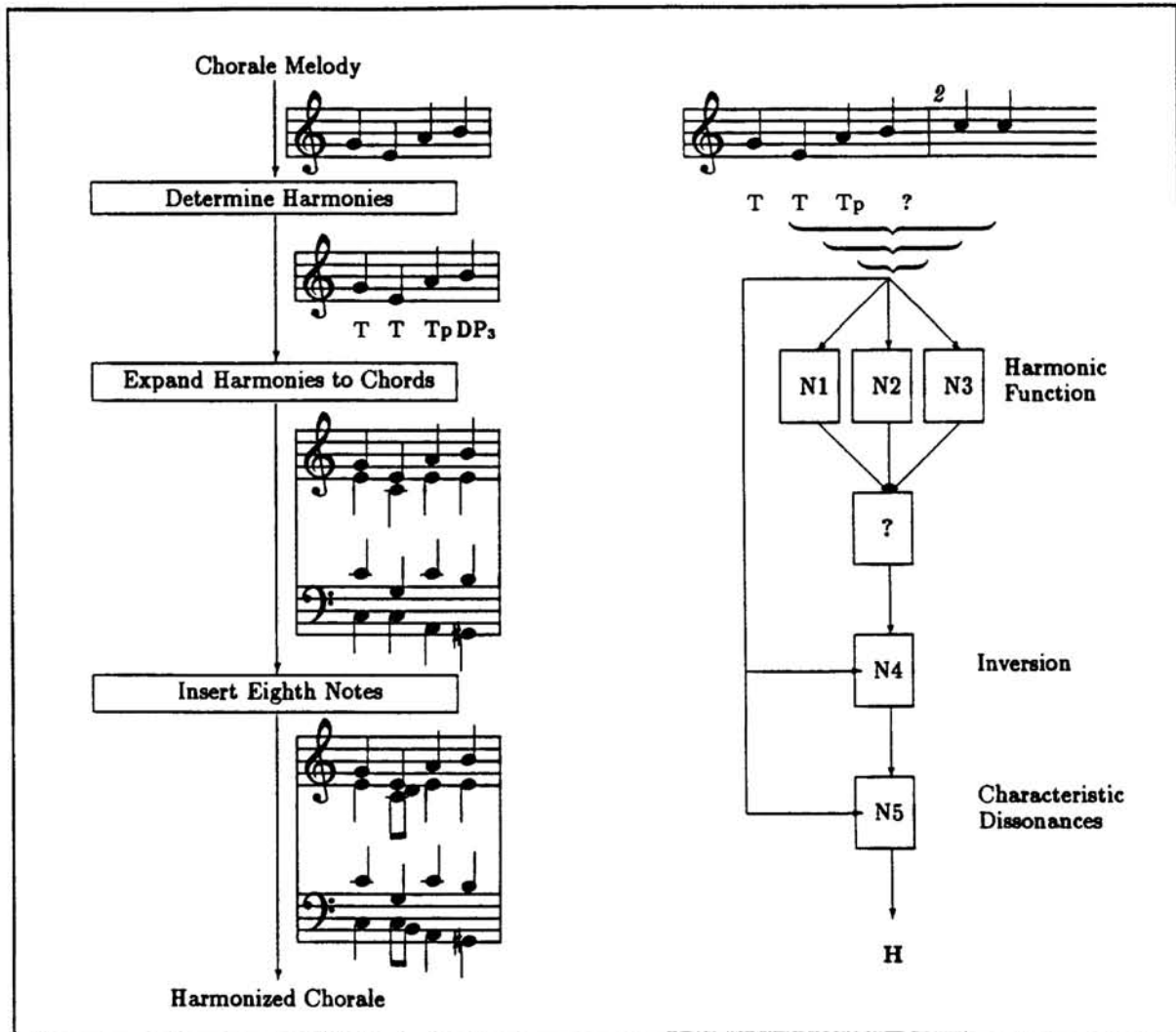

Figure 4: Left side: Overall structure of HARMONET. Right side: A more special-ized architecture with parallel and sequential nets (see text).

## 4    PERFORMANCE

HARMONET was trained separately on two sets of Bach chorales, each containing 20 chorales in major and minor keys, respectively. By passing the chorales through a window as explained above, each set amounted to approx. 1000 training examples. All nets were trained with the error backpropagation algorithm, needing 50 to 100 epochs to achieve reasonable convergence.

Figures 5 and 6 show two harmonizations produced by HARMONET, given melodies which were not in the training set. An audience of music professionals judged the quality of these and other chorales produced by HARMONET to be on the level on an improvising organist. HARMONET also compares well to non-neural ap-proaches. In figure 6 HARMONET's accompaniment is shown on a chorale melody also used in the Ph.D. thesis of (Ebcioglu, 1986) to demonstrate the expert system "CHORAL".

Figure 5: A chorale in a major key harmonized by HARMONET.

Figure 6: "Happy Birthday" harmonized by HARMONET.

## 5  CONCLUSIONS

The music processing system HARMONET presented in this paper clearly shows that musical real-world applications are well within the reach of connectionist approaches. We believe that HARMONET owes much of its success to a clean task decomposition and a meaningful selection and representation of musically relevant features. By using a hybrid approach we allow the networks to concentrate on musical essentials instead of on structural constraints which may be hard for a network to learn but easy to code symbolically. The abstraction of chords to harmonies reduces the problem space and resembles a musician's problem approach. The "harmonic representation" of pitch shows the harmonic character of the given melody more explicitly.

We have also experimented to replace the neural nets in HARMONET by other learning techniques such as decision trees (ID3) or nearest neighbor classification. However, as also reported on other tasks (Dietterich et al., 1990), they were outperformed by the neural nets.

HARMONET is not a general music processing system, its architecture is designed to solve a quite difficult but also quite specific task. However, due to HARMONET's neural learning component, only a comparatively small amount of musical expert knowledge was necessary to design the system, making it easier to build and more flexible than a pure rule based system.

### Acknowledgements

We thank Heinz Braun, Heiko Harms and Gudrun Socher for many fruitful discussions and contributions to this research and our music lab.

## Footnotes

[1]To be sure, $f$ is not a function but a relation, since there are many "legal" accompaniments for one melody. For simplicity, we view $f$ as a function.

### References

J.S.Bach (Ed.: Bernhard Friedrich Fischer) **389 Choralgesänge für vierstimmigen Chor**. Edition Breitkopf, Nr. 3765.

Dietterich,T.G., Hild,H., & Bakiri,G. **A comparative study of ID3 and Backpropagation for English Text-to-Speech Mapping**. Proc. of the Seventh International Conference on Machine Learning (pp. 24-31). Kaufmann, 1990.

Ebcioğlu,K. **An Expert System for Harmonization of Chorales in the Style of J.S.Bach**. Ph.D. Dissertation, Department of C.S., State University of New York at Buffalo, New York, 1986.

Lischka,C. **Understanding Music Cognition**. GMD St.Augustin, FRG, 1989.

Mozer,M.C., Soukup,T. **Connectionist Music Composition Based on Melodic and Stylistic Constraints**. Advances in Neural Information Processing 3 (NIPS 3), R.P. Lippmann, J. E. Moody, D.S. Touretzky (eds.), Kaufmann 1991.

Shepard, Roger N. **Geometrical Approximations to the Structure of Musical Pitch**. Psychological Review, Vol. 89, Nr. 4, July 1982.

Todd, Peter M. **A Connectionist Approach To Algorithmic Composition**. Computer Music Journal, Vol. 13, No. 4, Winter 1989.